# Fast Neural Network Emulation of Dynamical Systems for Computer Animation

**Radek Grzeszczuk** [1]    **Demetri Terzopoulos** [2]    **Geoffrey Hinton** [2]

[1] Intel Corporation
Microcomputer Research Lab
2200 Mission College Blvd.
Santa Clara, CA 95052, USA

[2] University of Toronto
Department of Computer Science
10 King's College Road
Toronto, ON  M5S 3H5, Canada

## Abstract

Computer animation through the numerical simulation of physics-based graphics models offers unsurpassed realism, but it can be computationally demanding. This paper demonstrates the possibility of replacing the numerical simulation of nontrivial dynamic models with a dramatically more efficient "NeuroAnimator" that exploits neural networks. NeuroAnimators are automatically trained off-line to emulate physical dynamics through the observation of physics-based models in action. Depending on the model, its neural network emulator can yield physically realistic animation one or two orders of magnitude faster than conventional numerical simulation. We demonstrate NeuroAnimators for a variety of physics-based models.

## 1  Introduction

Animation based on physical principles has been an influential trend in computer graphics for over a decade (see, e.g., [1, 2, 3]). This is not only due to the unsurpassed realism that physics-based techniques offer. In conjunction with suitable control and constraint mechanisms, physical models also facilitate the production of copious quantities of realistic animation in a highly automated fashion. Physics-based animation techniques are beginning to find their way into high-end commercial systems. However, a well-known drawback has retarded their broader penetration—compared to geometric models, physical models typically entail formidable numerical simulation costs.

This paper proposes a new approach to creating physically realistic animation that differs

radically from the conventional approach of numerically simulating the equations of motion of physics-based models. We replace physics-based models by fast *emulators* which automatically learn to produce similar motions by observing the models in action. Our emulators have a neural network structure, hence we dub them *NeuroAnimators*.

Our work is inspired in part by that of Nguyen and Widrow [4]. Their "truck backer-upper" demonstrated the neural network based approximation and control of a nonlinear kinematic system. We introduce several generalizations that enable us to tackle a variety of complex, fully dynamic models in the context of computer animation. Connectionist approximations of dynamical systems have been also been applied to robot control (see, e.g., [5, 6]).

## 2   The NeuroAnimator Approach

Our approach is motivated by the following considerations: Whether we are dealing with rigid [2], articulated [3], or nonrigid [1] dynamic animation models, the numerical simulation of the associated equations of motion leads to the computation of a discrete-time dynamical system of the form $s_{t+\delta t} = \Phi[s_t, u_t, f_t]$. These (generally nonlinear) equations express the vector $s_{t+\delta t}$ of state variables of the system (values of the system's degrees of freedom and their velocities) at time $t + \delta t$ in the future as a function $\Phi$ of the state vector $s_t$, the vector $u_t$ of control inputs, and the vector $f_t$ of external forces acting on the system at time $t$.

Physics-based animation through the numerical simulation of a dynamical system requires the evaluation of the map $\Phi$ at every timestep, which usually involves a non-trivial computation. Evaluating $\Phi$ using explicit time integration methods incurs a computational cost of $O(N)$ operations, where $N$ is proportional to the dimensionality of the state space. Unfortunately, for many dynamic models of interest, explicit methods are plagued by instability, necessitating numerous tiny timesteps $\delta t$ per unit simulation time. Alternatively, implicit time-integration methods usually permit larger timesteps, but they compute $\Phi$ by solving a system of $N$ algebraic equations, generally incurring a cost of $O(N^3)$ per timestep.

Is it possible to replace the conventional numerical simulator by a significantly cheaper alternative? A crucial realization is that the substitute, or emulator, need not compute the map $\Phi$ exactly, but merely approximate it to a degree of precision that preserves the perceived faithfulness of the resulting animation to the simulated dynamics of the physical model. Neural networks offer a general mechanism for approximating complex maps in higher dimensional spaces [7].[1] Our premise is that, to a sufficient degree of accuracy and at significant computational savings, trained neural networks can approximate maps $\Phi$ not just for simple dynamical systems, but also for those associated with dynamic models that are among the most complex reported in the graphics literature to date.

The NeuroAnimator, which uses neural networks to emulate physics-based animation, learns an approximation to the dynamic model by observing instances of state transitions, as well as control inputs and/or external forces that cause these transitions. By generalizing from the sparse examples presented to it, a trained NeuroAnimator can emulate an infinite variety of continuous animations that it has never actually seen. Each emulation step costs only $O(N^2)$ operations, but it is possible to gain additional efficiency relative to a numerical simulator by training neural networks to approximate a lengthy chain of evaluations of the discrete-time dynamical system. Thus, the emulator network can perform "super

timesteps" $\Delta t = n\delta t$, typically one or two orders of magnitude larger than $\delta t$ for the competing implicit time-integration scheme, thereby achieving outstanding efficiency without serious loss of accuracy.

## 3   From Physics-Based Models to NeuroAnimators

Our task is to construct neural networks that approximate $\Phi$ in the dynamical system. We propose to employ backpropagation to train feedforward networks $\mathbf{N}_\Phi$, with a single layer of sigmoidal hidden units, to predict future states using super timesteps $\Delta t = n\delta t$ while containing the approximation error so as not to appreciably degrade the physical realism of the resulting animation. The basic emulation step is $\mathbf{s}_{t+\Delta t} = \mathbf{N}_\Phi[\mathbf{s}_t, \mathbf{u}_t, \mathbf{f}_t]$. The trained emulator network $\mathbf{N}_\Phi$ takes as input the state of the model, its control inputs, and the external forces acting on it at time $t$, and produces as output the state of the model at time $t + \Delta t$ by evaluating the network. The emulation process is a sequence of these evaluations. After each evaluation, the network control and force inputs receive new values, and the network state inputs receive the emulator outputs from the previous evaluation. Since the emulation step is large compared with the numerical simulation step, we resample the motion trajectory at the animation frame rate, computing intermediate states through linear interpolation of states obtained from the emulation.

### 3.1   Network Input/Output Structure

Fig. 1(a) illustrates different emulator input/output structures. The emulator network has a single set of output variables specifying $\mathbf{s}_{t+\Delta t}$. In general, for a so-called active model, which includes control inputs, under the influence of unpredictable applied forces, we employ a full network with three sets of input variables: $\mathbf{s}_t$, $\mathbf{u}_t$, and $\mathbf{f}_t$, as shown in the figure. For passive models, the control $\mathbf{u}_t = \mathbf{0}$ and the network simplifies to one with two sets of inputs, $\mathbf{s}_t$ and $\mathbf{f}_t$. In the special case when the forces $\mathbf{f}_t$ are completely determined by the state of the system $\mathbf{s}_t$, we can suppress the $\mathbf{f}_t$ inputs, allowing the network to learn the effects of these forces from the state transition training data, thus yielding a simpler emulator with two input sets $\mathbf{s}_t$ and $\mathbf{u}_t$. The simplest type of emulator has only a single set of inputs $\mathbf{s}_t$. This emulator suffices to approximate passive models acted upon by deterministic external forces.

### 3.2   Input and Output Transformations

The accurate approximation of complex functional mappings using neural networks can be challenging. We have observed that a simple feedforward neural network with a single layer of sigmoid units has difficulty producing an accurate approximation to the dynamics of physical models. In practice, we often must transform the emulator to ensure a good approximation of the map $\Phi$.

A fundamental problem is that the state variables of a dynamical system can have a large dynamic range (in principle, from $-\infty$ to $+\infty$). To approximate a nonlinear map $\Phi$ accurately over a large domain, we would need to use a neural network with many sigmoid units, each shifted and scaled so that their nonlinear segments cover different parts of the domain. The direct approximation of $\Phi$ is therefore impractical. A successful strategy is to train networks to emulate *changes* in state variables rather than their actual values, since state changes over small timesteps will have a significantly smaller dynamic range. Hence, in Fig. 1(b) (top) we restructure our simple network $\mathbf{N}_\Phi$ as a network $\mathbf{N}_\Phi^\Delta$ which is trained

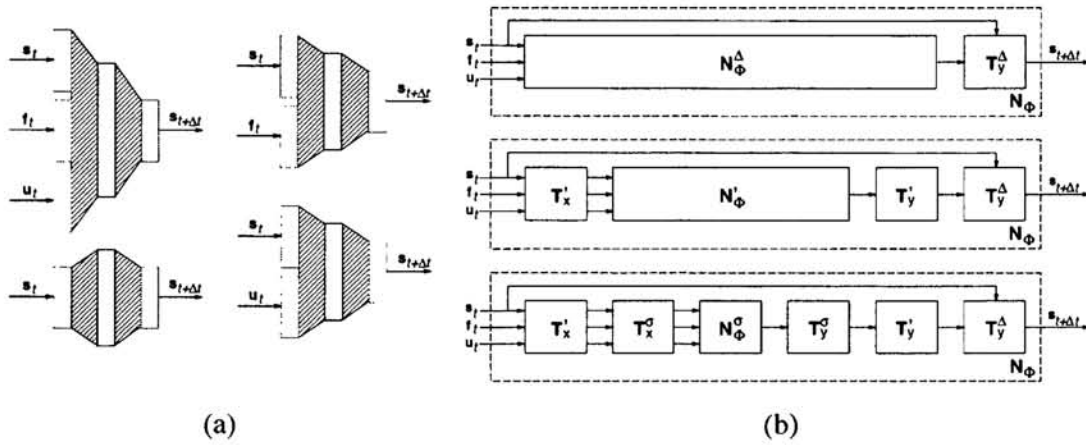

<div style="text-align:center">

(a)                     (b)

</div>

Figure 1: (a) Different types of emulators. (b) Transforming a simple feedforward neural network $N_\Phi$ into a practical emulator network $N_\Phi^\sigma$ that is easily trained to emulate physics-based models. The following operators perform the appropriate pre- and post-processing: $T_x'$ transforms inputs to local coordinates, $T_x^\sigma$ normalizes inputs, $T_y^\sigma$ unnormalizes outputs, $T_y'$ transforms outputs to global coordinates, $T_y^\Delta$ converts from a state change to the next state (see text and [8] for the details).

to emulate the change in the state vector $\Delta s_t$ for given state, external force, and control inputs, followed by an operator $T_y^\Delta$ that computes $s_{t+\Delta t} = s_t + \Delta s_t$ to recover the next state.

We can further improve the approximation power of the emulator network by exploiting natural invariances. In particular, since the map $\Phi$ is invariant under rotation and translation, we replace $N_\Phi^\Delta$ with an operator $T_x'$ that converts the inputs from the world coordinate system to the local coordinate system of the model, a network $N_\Phi'$ that is trained to emulate state changes represented in the local coordinate system, and an operator $T_y'$ that converts the output of $N_\Phi'$ back to world coordinates (Fig. 1(b) (center)).

Since the values of state, force, and control variables can deviate significantly, their effect on the network outputs is uneven, causing problems when large inputs must have a small influence on outputs. To make inputs contribute more evenly to the network outputs, we normalize groups of variables so that they have zero means and unit variances. With normalization, we can furthermore expect the weights of the trained network to be of order unity and they can be given a simple random initialization prior to training. Hence, in Fig. 1(b)) (bottom) we replace $N_\Phi'$ with an operator $T_x^\sigma$ that normalizes its inputs, a network $N_\Phi^\sigma$ that assumes zero mean, unit variance inputs and outputs, and an operator $T_y^\sigma$ that unnormalizes the outputs to recover their original distributions.

Although the final emulator in Fig. 1(b) is structurally more complex than the standard feedforward neural network $N_\Phi$ that it replaces, the operators denoted by $T$ are completely determined by the state of the model and the distribution of the training data, and the emulator network $N_\Phi^\sigma$ is much easier to train.

## 3.3 Hierarchical Networks

As a universal function approximator, a neural network should in principle be able to approximate the map $\Phi$ for any dynamical system, given enough sigmoid hidden units and

training data. In practice, however, the number of hidden layer neurons needed and the training data requirements grow quickly with the size of the network, often making the training of large networks impractical. To overcome the "curse of dimensionality," we have found it prudent to structure NeuroAnimators for all but the simplest physics-based models as hierarchies of smaller networks rather than as large, monolithic networks. The strategy behind a hierarchical representation is to group state variables according to their dependencies and approximate each tightly coupled group with a subnet that takes part of its input from a parent network.

### 3.4 Training NeuroAnimators

To arrive at a NeuroAnimator for a given physics-based model, we train the constituent neural network(s) through backpropagation on training examples generated by simulating the model. Training requires the generation and processing of many examples, hence it is typically slow, often requiring several CPU hours. However, once a NeuroAnimator is trained offline, it can be reused online to produce an infinite variety of fast animations. The important point is that by generalizing from the sparse training examples, a trained NeuroAnimator will produce an infinite variety of extended, continuous animations that it has never "seen".

More specifically, each training example consists of an input vector $\mathbf{x}$ and an output vector $\mathbf{y}$. In the general case, the input vector $\mathbf{x} = [\mathbf{s}_0^T, \mathbf{f}_0^T, \mathbf{u}_0^T]^T$ comprises the state of the model, the external forces, and the control inputs at time $t = 0$. The output vector $\mathbf{y} = \mathbf{s}_{\Delta t}$ is the state of the model at time $t = \Delta t$, where $\Delta t$ is the duration of the super timestep. To generate each training example, we could start the numerical simulator of the physics-based model with the initial conditions $\mathbf{s}_0$, $\mathbf{f}_0$, and $\mathbf{u}_0$, and run the dynamic simulation for $n$ numerical time steps $\delta t$ such that $\Delta t = n\delta t$. In principle, we could generate an arbitrarily large set of training examples $\{\mathbf{x}^\tau; \mathbf{y}^\tau\}$, $\tau = 1, 2, \ldots$, by repeating this process with different initial conditions. To learn a good neural network approximation $\mathbf{N}_\Phi$ of the map $\Phi$, we would like ideally to sample $\Phi$ as uniformly as possible over its domain, with randomly chosen initial conditions among all valid state, external force, and control combinations. However, we can make better use of computational resources by sampling those state, force, and control inputs that typically occur as a physics-based model is used in practice.

We employ a neural network simulator called *Xerion* which was developed at the University of Toronto. We begin the off-line training process by initializing the weights of $\mathbf{N}_\Phi^\sigma$ to random values from a uniform distribution in the range $[0, 1]$ (due to the normalization of inputs and outputs). Xerion automatically terminates the backpropagation learning algorithm when it can no longer reduce the network approximation error significantly. We use the conjugate gradient method to train networks of small and moderate size. For large networks, we use gradient descent with momentum. We divide the training examples into mini-batches, each consisting of approximately 30 uncorrelated examples, and update the network weights after processing each mini-batch.

## 4   Results

We have successfully constructed and trained several NeuroAnimators to emulate a variety of physics-based models (Fig. 2). We used SD/FAST (a rigid body dynamics simulator marketed by Symbolic Dynamics, Inc.) to simulate the dynamics of the rigid body

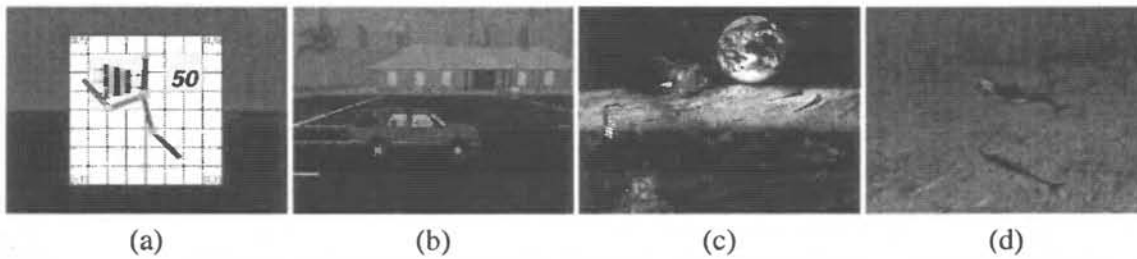

$$(a) \qquad\qquad (b) \qquad\qquad (c) \qquad\qquad (d)$$

Figure 2: NeuroAnimators used in our experiments. (a) Emulator of a physics-based model of a planar multi-link pendulum suspended in gravity, subject to joint friction forces, external forces applied on the links, and controlled by independent motor torques at each of the three joints. (b) Emulator of a physics-based model of a truck implemented as a rigid body, subject to friction forces where the tires contact the ground, controlled by rear-wheel drive (forward and reverse) and steerable front wheels. (c) Emulator of a physics-based model of a lunar lander, implemented as a rigid body subject to gravitational forces and controlled by a main rocket thruster and three independent attitude jets. (d) Emulator of a biomechanical (mass-spring-damper) model of a dolphin capable of swimming in simulated water via the coordinated contraction of 6 independently controlled muscle actuators which deform its body, producing hydrodynamic propulsion forces.

and articulated models, and we employ the simulator developed in [10] to simulate the deformable-body dynamics of the dolphin.

In our experiments we have not attempted to minimize the number of network weights required for successful training. We have also not tried to minimize the number of sigmoidal hidden units, but rather used enough units to obtain networks that generalize well while not overfitting the training data. We can always expect to be able to satisfy these guidelines in view of our ability to generate sufficient training data.

An important advantage of using neural networks to emulate dynamical systems is the speed at which they can be iterated to produce animation. Since the emulator for a dynamical system with the state vector of size $N$ never uses more than $O(N)$ hidden units, it can be evaluated using only $O(N^2)$ operations. By comparison, a single simulation timestep using an implicit time integration scheme requires $O(N^3)$ operations. Moreover, a forward pass through the neural network is often equivalent to as many as 50 physical simulation steps, so the efficiency is even more dramatic, yielding performance improvements up to two orders of magnitude faster than the physical simulator. A NeuroAnimator that predicts 100 physical simulation steps offers a speedup of anywhere between 50 and 100 times depending on the type of physical model.

## 5  Control Learning

An additional benefit of the NeuroAnimator is that it enables a novel, highly efficient approach to the difficult problem of controlling physics-based models to synthesize motions that satisfy prescribed animation goals. The neural network approximation to the physical model is differentiable; hence, it can be used to discover the causal effects that control force inputs have on the actions of the models. Outstanding efficiency stems from exploiting the trained NeuroAnimator to compute partial derivatives of output states with respect to control inputs. The efficient computation of the approximate gradient enables the utilization of fast gradient-based optimization for controller synthesis.

Nguyen and Widrow's [4] "truck backer-upper" demonstrated the neural network based approximation and control of a nonlinear kinematic system. Our technique offers a new controller synthesis algorithm that works well in dynamic environments with changing control objectives. See [8, 9] for the details.

## 6   Conclusion

We have introduced an efficient alternative to the conventional approach of producing physically realistic animation through numerical simulation. Our approach involves the learning of neural network emulators of physics-based models by observing the dynamic state transitions produced by such models in action. The emulators approximate physical dynamics with dramatic efficiency, yet without serious loss of apparent fidelity. Our performance benchmarks indicate that the neural network emulators can yield physically realistic animation one or two orders of magnitude faster than conventional numerical simulation of the associated physics-based models. Our new control learning algorithm, which exploits fast emulation and the differentiability of the network approximation, is orders of magnitude faster than competing controller synthesis algorithms for computer animation.

### Acknowledgements

We thank Zoubin Ghahramani for valuable discussions leading to the idea of the rotation and translation invariant emulator, which was crucial to the success of this work. We are indebted to Steve Hunt, John Funge, Alexander Reshetov, Sonja Jeter and Mike Gendimenico at Intel, and Mike Revow, Drew van Camp and Michiel van de Panne at the University of Toronto for their assistance.

## Footnotes

[1] Note that $\Phi$ is in general a high-dimensional map from $\Re^{s+u+f} \mapsto \Re^s$, where $s$, $u$, and $f$ denote the dimensionalities of the state, control, and external force vectors.

## References

[1]  D. Terzopoulos, J. Platt, A. Barr, K. Fleischer. Elastically deformable models. In M.C. Stone, ed., *Computer Graphics (SIGGRAPH '87 Proceedings)*, **21**, 205–214, July 1987.

[2]  J.K. Hahn. Realistic animation of rigid bodies. In J. Dill, ed., *Computer Graphics (SIGGRAPH '88 Proceedings)*, **22**, 299–308, August 1988.

[3]  J.K. Hodgins, W.L. Wooten, D.C. Brogan, J.F. O'Brien. Animating human athletics. In R. Cook, ed., *Proc. of ACM SIGGRAPH 95 Conf.*, 71–78, August, 1995.

[4]  D. Nguyen, B. Widrow. The truck backer-upper: An example of self-learning in neural networks. In *Proc. Inter. Joint Conf. Neural Networks*, 357–363. IEEE Press, 1989.

[5]  M. I. Jordan. Supervised learning and systems with excess degrees of freedom. Technical Report 88-27, Univ. of Massachusetts, Comp.& Info. Sci., Amherst, MA, 1988.

[6]  K. S. Narendra, K. Parthasarathy. Gradient methods for the optimization of dynamical systems containing neural networks. *IEEE Trans. on Neural Networks*, 2(2):252–262, 1991.

[7]  G. Cybenko. Approximation by superposition of sigmoidal function. *Math. of Control Signals & Systems*, 2(4):303–314, 1989.

[8]  R. Grzeszczuk. *NeuroAnimator: Fast Neural Network Emulation and Control of Physics-Based Models*. PhD thesis, Dept. of Comp. Sci., Univ. of Toronto, May 1998.

[9]  R. Grzeszczuk, D. Terzopoulos, G. Hinton. NeuroAnimator: Fast neural network emulation and control of physics-based models. In M. Cohen, ed., *Proc. of ACM SIGGRAPH 98 Conf.*, 9–20, July 1998.

[10] X. Tu, D. Terzopoulos. Artificial fishes: Physics, locomotion, perception, behavior. In A. Glassner, ed., *Proc. of ACM SIGGRAPH 94 Conf.*, 43–50. July 1994.